# Interpolating Between Types and Tokens by Estimating Power-Law Generators *

**Sharon Goldwater**     **Thomas L. Griffiths**     **Mark Johnson**
Department of Cognitive and Linguistic Sciences
Brown University, Providence RI 02912, USA
{sharon_goldwater,tom_griffiths,mark_johnson}@brown.edu

## Abstract

Standard statistical models of language fail to capture one of the most striking properties of natural languages: the power-law distribution in the frequencies of word tokens. We present a framework for developing statistical models that generically produce power-laws, augmenting standard generative models with an *adaptor* that produces the appropriate pattern of token frequencies. We show that taking a particular stochastic process – the Pitman-Yor process – as an adaptor justifies the appearance of type frequencies in formal analyses of natural language, and improves the performance of a model for unsupervised learning of morphology.

## 1   Introduction

In general it is important for models used in unsupervised learning to be able to describe the gross statistical properties of the data they are intended to learn from, otherwise these properties may distort inferences about the parameters of the model. One of the most striking statistical properties of natural languages is that the distribution of word frequencies is closely approximated by a power-law. That is, the probability that a word $w$ will occur with frequency $n_w$ in a sufficiently large corpus is proportional to $n_w^{-g}$. This observation, which is usually attributed to Zipf [1] but enjoys a long and detailed history [2], stimulated intense research in the 1950s (e.g., [3]) but has largely been ignored in modern computational linguistics. By developing models that generically exhibit power-laws, it may be possible to improve methods for unsupervised learning of linguistic structure.

In this paper, we introduce a framework for developing generative models for language that produce power-law distributions. Our framework is based upon the idea of specifying language models in terms of two components: a *generator*, an underlying generative model for words which need not (and usually does not) produce a power-law distribution, and an *adaptor*, which transforms the stream of words produced by the generator into one whose frequencies obey a power law distribution. This framework is extremely general: any generative model for language can be used as a generator, with the power-law distribution being produced as the result of making an appropriate choice for the adaptor.

In our framework, estimation of the parameters of the generator will be affected by assumptions about the form of the adaptor. We show that use of a particular adaptor, the Pitman-Yor process [4, 5, 6], sheds light on a tension exhibited by formal approaches to natural language: whether explanations should be based upon the *types* of words that languages

*This work was partially supported by NSF awards IGERT 9870676 and ITR 0085940 and NIMH award 1R0-IMH60922-01A2

exhibit, or the frequencies with which *tokens* of those words occur. One place where this tension manifests is in accounts of morphology, where formal linguists develop accounts of why particular words appear in the lexicon (e.g., [7]), while computational linguists focus on statistical models of the frequencies of tokens of those words (e.g., [8]). The tension between types and tokens also appears within computational linguistics. For example, one of the most successful forms of smoothing used in statistical language models, Kneser-Ney smoothing, explicitly interpolates between type and token frequencies [9, 10, 11].

The plan of the paper is as follows. Section 2 discusses stochastic processes that can produce power-law distributions, including the Pitman-Yor process. Section 3 specifies a two-stage language model that uses the Pitman-Yor process as an adaptor, and examines some properties of this model: Section 3.1 shows that estimation based on type and token frequencies are special cases of this two-stage language model, and Section 3.2 uses these results to provide a novel justification for the use of Kneser-Ney smoothing. Section 4 describes a model for unsupervised learning of the morphological structure of words that uses our framework, and demonstrates that its performance improves as we move from estimation based upon tokens to types. Section 5 concludes the paper.

## 2 Producing power-law distributions

Assume we want to generate a sequence of $N$ outcomes, $\mathbf{z} = \{z_1, \ldots, z_N\}$ with each outcome $z_i$ being drawn from a set of (possibly unbounded) size $Z$. Many of the stochastic processes that produce power-laws are based upon the principle of *preferential attachment*, where the probability that the $i$th outcome, $z_i$, takes on a particular value $k$ depends upon the frequency of $k$ in $\mathbf{z}_{-i} = \{z_1, \ldots, z_{i-1}\}$ [2]. For example, one of the earliest and most widely used preferential attachment schemes [3] chooses $z_i$ according to the distribution

$$P(z_i = k \mid \mathbf{z}_{-i}) = a\frac{1}{Z} + (1-a)\frac{n_k^{(\mathbf{z}_{-i})}}{i-1} \tag{1}$$

where $n_k^{(\mathbf{z}_{-i})}$ is the number of times $k$ occurs in $\mathbf{z}_{-i}$. This "rich-get-richer" process means that a few outcomes appear with very high frequency in $\mathbf{z}$ – the key attribute of a power-law distribution. In this case, the power-law has parameter $g = 1/(1-a)$.

One problem with these classical models is that they assume a fixed ordering on the outcomes $\mathbf{z}$. While this may be appropriate for some settings, the assumption of a temporal ordering restricts the contexts in which such models can be applied. In particular, it is much more restrictive than the assumption of independent sampling that underlies most statistical language models. Consequently, we will focus on a different preferential attachment scheme, based upon the two-parameter species sampling model [4, 5] known as the Pitman-Yor process [6]. Under this scheme outcomes follow a power-law distribution, but remain *exchangeable*: the probability of a set of outcomes is not affected by their ordering.

The Pitman-Yor process can be viewed as a generalization of the Chinese restaurant process [6]. Assume that $N$ customers enter a restaurant with infinitely many tables, each with infinite seating capacity. Let $z_i$ denote the table chosen by the $i$th customer. The first customer sits at the first table, $z_1 = 1$. The $i$th customer chooses table $k$ with probability

$$P(z_i = k \mid \mathbf{z}_{-i}) = \begin{cases} \frac{n_k^{(\mathbf{z}_{-i})} - a}{i-1+b} & k \leq K(\mathbf{z}_{-i}) \\ \frac{K(\mathbf{z}_{-i})a + b}{i-1+b} & k = K(\mathbf{z}_{-i}) + 1 \end{cases} \tag{2}$$

where $a$ and $b$ are the two parameters of the process and $K(\mathbf{z}_{-i})$ is the number of tables that are currently occupied.

The Pitman-Yor process satisfies our need for a process that produces power-laws while retaining exchangeability. Equation 2 is clearly a preferential attachment scheme. When

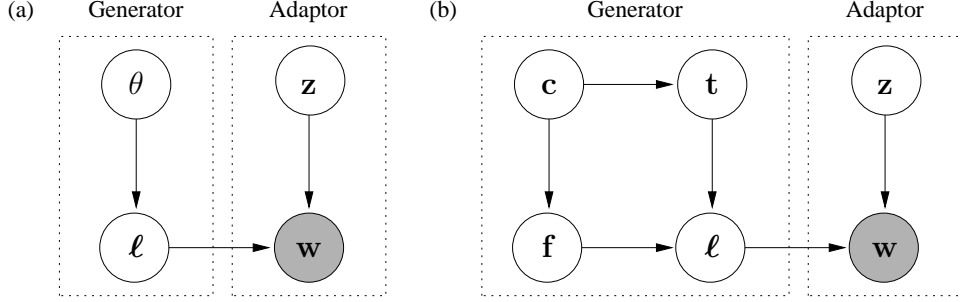

Figure 1: Graphical models showing dependencies among variables in (a) the simple two-stage model, and (b) the morphology model. Shading of the node containing $\mathbf{w}$ reflects the fact that this variable is observed. Dotted lines delimit the generator and adaptor.

$a = 0$ and $b > 0$, it reduces to the standard Chinese restaurant process [12, 4] used in Dirichlet process mixture models [13]. When $0 < a < 1$, the number of people seated at each table follows a power-law distribution with $g = 1 + a$ [5]. It is straightforward to show that the customers are exchangeable: the probability of a partition of customers into sets seated at different tables is unaffected by the order in which the customers were seated.

## 3  A two-stage language model

We can use the Pitman-Yor process as the foundation for a language model that generically produces power-law distributions. We will define a two-stage model by extending the restaurant metaphor introduced above. Imagine that each table $k$ is labelled with a word $\ell_k$ from a vocabulary of (possibly unbounded) size $W$. The first stage is to generate these labels, sampling $\ell_k$ from a generative model for words that we will refer to as the *generator*. For example, we could choose to draw the labels from a multinomial distribution $\theta$. The second stage is to generate the actual sequence of words itself. This is done by allowing a sequence of customers to enter the restaurant. Each customer chooses a table, producing a seating arrangement, $\mathbf{z}$, and says the word used to label that the table, producing a sequence of words, $\mathbf{w}$. The process by which customers choose tables, which we will refer to as the *adaptor*, defines a probability distribution over the sequence of words $\mathbf{w}$ produced by the customers, determining the frequency with which tokens of the different types occur. The statistical dependencies among the variables in one such model are shown in Figure 1 (a).

Given the discussion in the previous section, the Pitman-Yor process is a natural choice for an adaptor. The result is technically a Pitman-Yor mixture model, with $z_i$ indicating the "class" responsible for generating the $i$th word, and $\ell_k$ determining the multinomial distribution over words associated with class $k$, with $P(w_i = w \mid z_i = k, \ell_k) = 1$ if $\ell_k = w$, and 0 otherwise. Under this model the probability that the $i$th customer produces word $w$ given previously produced words $\mathbf{w}_{-i}$ and current seating arrangement $\mathbf{z}_{-i}$ is

$$P(w_i = w \mid \mathbf{w}_{-i}, \mathbf{z}_{-i}, \theta) = \sum_k \sum_{\ell_k} P(w_i = w \mid z_i = k, \ell_k) P(\ell_k \mid \mathbf{w}_{-i}, \mathbf{z}_{-i}, \theta) P(z_i = k \mid \mathbf{z}_{-i})$$

$$= \sum_{k=1}^{K(\mathbf{z}_{-i})} \frac{n_k^{(\mathbf{z}_{-i})} - a}{i - 1 + b} I(\ell_k = w) + \frac{K(\mathbf{z}_{-i})a + b}{i - 1 + b} \theta_w \tag{3}$$

where $I(\cdot)$ is an indicator function, being 1 when its argument is true and 0 otherwise. If $\theta$ is uniform over all $W$ words, then the distribution over $\mathbf{w}$ reduces to the Pitman-Yor process as $W \to \infty$. Otherwise, multiple tables can receive the same label, increasing the frequency of the corresponding word and producing a distribution with $g < 1 + a$. Again, it is straightforward to show that words are exchangeable under this distribution.

### 3.1 Types and tokens

The use of the Pitman-Yor process as an adaptor provides a justification for the role of word types in formal analyses of natural language. This can be seen by considering the question of how to estimate the parameters of the multinomial distribution used as a generator, $\theta$.[1] In general, the parameters of generators can be estimated using Markov chain Monte Carlo methods, as we demonstrate in Section 4. In this section, we will show that estimation schemes based upon type and token frequencies are special cases of our language model, corresponding to the extreme values of the parameter $a$. Values of $a$ between these extremes identify estimation methods that interpolate between types and tokens.

Taking a multinomial distribution with parameters $\theta$ as a generator and the Pitman-Yor process as an adaptor, the probability of a sequence of words $\mathbf{w}$ given $\theta$ is

$$P(\mathbf{w}\,|\,\theta) = \sum_{\mathbf{z},\boldsymbol{\ell}} P(\mathbf{w},\mathbf{z},\boldsymbol{\ell}\,|\,\theta) = \sum_{\mathbf{z},\boldsymbol{\ell}} \frac{\Gamma(b)}{\Gamma(N+b)} \prod_{k=1}^{K(\mathbf{z})} \left( \theta_{\ell_k}((k-1)a+b)\frac{\Gamma(n_k^{(\mathbf{z})}-a)}{\Gamma(1-a)} \right)$$

where in the last sum $\mathbf{z}$ and $\boldsymbol{\ell}$ are constrained such that $\ell_{z_i} = w_i$ for all $i$. In the case where $b = 0$, this simplifies to

$$P(\mathbf{w}\,|\,\theta) = \sum_{\mathbf{z},\boldsymbol{\ell}} \left( \prod_{k=1}^{K(\mathbf{z})} \theta_{\ell_k} \right) \cdot \frac{\Gamma(K(\mathbf{z}))}{\Gamma(N)} \cdot a^{K(\mathbf{z})-1} \cdot \left( \prod_{k=1}^{K(\mathbf{z})} \frac{\Gamma(n_k^{(\mathbf{z})}-a)}{\Gamma(1-a)} \right) \qquad (4)$$

The distribution $P(\mathbf{w}\,|\,\theta)$ determines how the data $\mathbf{w}$ influence estimates of $\theta$, so we will consider how $P(\mathbf{w}\,|\,\theta)$ changes under different limits of $a$.

In the limit as $a$ approaches 1, estimation of $\theta$ is based upon word tokens. When $a \to 1$, $\frac{\Gamma(n_k^{\mathbf{z}}-a)}{\Gamma(1-a)}$ is 1 for $n_k^{(\mathbf{z})} = 1$ but approaches 0 for $n_k^{(\mathbf{z})} > 1$. Consequently, all terms in the sum over $(\mathbf{z},\boldsymbol{\ell})$ go to zero, except that in which every word token has its own table. In this case, $K(\mathbf{z}) = N$ and $\ell_k = w_k$. It follows that $\lim_{a \to 1} P(\mathbf{w}\,|\,\theta) = \prod_{k=1}^{N} \theta_{w_k}$. Any form of estimation using $P(\mathbf{w}\,|\,\theta)$ will thus be based upon the frequencies of word tokens in $\mathbf{w}$.

In the limit as $a$ approaches 0, estimation of $\theta$ is based upon word types. The appearance of $a^{K(\mathbf{z})-1}$ in Equation 4 means that as $a \to 0$, the sum over $\mathbf{z}$ is dominated by the seating arrangement that minimizes the total number of tables. Under the constraint that $\ell_{z_i} = w_i$ for all $i$, this minimal configuration is the one in which every word type receives a single table. Consequently, $\lim_{a \to 0} P(\mathbf{w}\,|\,\theta)$ is dominated by a term in which there is a single instance of $\theta_w$ for each word $w$ that appears in $\mathbf{w}$.[2] Any form of estimation using $P(\mathbf{w}\,|\,\theta)$ will thus be based upon a single instance of each word type in $\mathbf{w}$.

### 3.2 Predictions and smoothing

In addition to providing a justification for the role of types in formal analyses of language in general, use of the Pitman-Yor process as an adaptor can be used to explain the assumptions behind a specific scheme for combining token and type frequencies: Kneser-Ney smoothing. Smoothing methods are schemes for regularizing empirical estimates of the probabilities of words, with the goal of improving the predictive performance of language models. The Kneser-Ney smoother estimates the probability of a word by combining type and token frequencies, and has proven particularly effective for $n$-gram models [9, 10, 11].

To use an $n$-gram language model, we need to estimate the probability distribution over words given their *history*, i.e. the $n$ preceding words. Assume we are given a vector of $N$ words $\mathbf{w}$ that all share a common history, and want to predict the next word, $w_{N+1}$, that will occur with that history. Assume that we also have vectors of words from $H$ other histories, $\mathbf{w}^{(1)}, \ldots, \mathbf{w}^{(H)}$. The interpolated Kneser-Ney smoother [11] makes the prediction

$$P(w_{N+1} = w \mid \mathbf{w}) = \frac{n_w^{(\mathbf{w})} - I(n_w^{(\mathbf{w})} > D)D}{N} + \frac{\sum_w I(n_w^{(\mathbf{w})} > D)D}{N} \frac{\sum_h I(w \in \mathbf{w}^{(h)})}{\sum_w \sum_h I(w \in \mathbf{w}^{(h)})} \quad (5)$$

where we have suppressed the dependence on $\mathbf{w}^{(1)}, \ldots, \mathbf{w}^{(H)}$, $D$ is a "discount factor" specified as a parameter of the model, and the sum over $h$ includes $\mathbf{w}$.

We can define a two-stage model appropriate for this setting by assuming that the sets of words for all histories are produced by the same adaptor and generator. Under this model, the probability of word $w_{N+1}$ given $\mathbf{w}$ and $\theta$ is

$$P(w_{N+1} = w \mid \mathbf{w}, \theta) = \sum_{\mathbf{z}} P(w_{N+1} = w | \mathbf{w}, \mathbf{z}, \theta) P(\mathbf{z} | \mathbf{w}, \theta)$$

where $P(w_{N+1} = w | \mathbf{w}, \mathbf{z}, \theta)$ is given by Equation 3. Assuming $b = 0$, this becomes

$$P(w_{N+1} = w \mid \mathbf{w}, \theta) = \frac{n_w^{\mathbf{w}} - E_{\mathbf{z}}[K_w(\mathbf{z})]\, a}{N} + \frac{\sum_w E_{\mathbf{z}}[K_w(\mathbf{z})]\, a}{N} \theta_w \quad (6)$$

where $E_{\mathbf{z}}[K_w(\mathbf{z})] = \sum_{\mathbf{z}} K_w(\mathbf{z}) P(\mathbf{z}|\mathbf{w}, \theta)$, and $K_w(\mathbf{z})$ is the number of tables with label $w$ under the seating assignment $\mathbf{z}$. The other histories enter into this expression via $\theta$. Since the words associated with each history is assumed to be produced from a single set of parameters $\theta$, the maximum-likelihood estimate of $\theta_w$ will approach

$$\theta_w = \frac{\sum_h I(w \in \mathbf{w}^{(h)})}{\sum_w \sum_h I(w \in \mathbf{w}^{(h)})}$$

as $a$ approaches 0, since only a single instance of each word type in each context will contribute to the estimate of $\theta$. Substituting this value of $\theta_w$ into Equation 6 reveals the correspondence to the Kneser-Ney smoother (Equation 5). The only difference is that the constant discount factor $D$ is replaced by $aE_{\mathbf{z}}[K_w(\mathbf{z})]$, which will increase slowly as $n_w$ increases. This difference might actually lead to an improved smoother: the Kneser-Ney smoother seems to produce better performance when $D$ increases as a function of $n_w$ [11].

## 4   Types and tokens in modeling morphology

Our attempt to develop statistical models of language that generically produce power-law distributions was motivated by the possibility that models that account for this statistical regularity might be able to learn linguistic information better than those that do not. Our two-stage language modeling framework allows us to create exactly these sorts of models, with the generator producing individual lexical items, and the adaptor producing the power-law distribution over words. In this section, we show that taking a generative model for morphology as the generator and varying the parameters of the adaptor results in an improvement in unsupervised learning of the morphological structure of English.

### 4.1   A generative model for morphology

Many languages contain words built up of smaller units of meaning, or *morphemes*. These units can contain lexical information (as stems) or grammatical information (as affixes). For example, the English word *walked* can be parsed into the stem *walk* and the past-tense suffix *ed*. Knowledge of morphological structure enables language learners to understand and produce novel wordforms, and facilitates tasks such as stemming (e.g., [14]).

As a basic model of morphology, we assume that each word consists of a single stem and suffix, and belongs to some inflectional class. Each class is associated with a stem distribution and a suffix distribution. We assume that stems and suffixes are independent given the class, so we have

$$P(\ell_k = w) = \sum_{c,t,f} I(w = t.f)P(c_k = c)P(t_k = t \mid c_k = c)P(f_k = f \mid c_k = c) \quad (7)$$

where $c_k$, $t_k$, and $f_k$ are the class, stem, and suffix associated with $\ell_k$, and $t.f$ indicates the concatenation of $t$ and $f$. In other words, we generate a label by first drawing a class, then drawing a stem and a suffix conditioned on the class. Each of these draws is from a multinomial distribution, and we will assume that these multinomials are in turn generated from symmetric Dirichlet priors, with parameters $\kappa$, $\tau$, and $\phi$ respectively. The resulting generative model can be used as the generator in a two-stage language model, providing a more structured replacement for the multinomial distribution, $\theta$. As before, we will use the Pitman-Yor process as an adaptor, setting $b = 0$. Figure 1 (b) illustrates the dependencies between the variables in this model.

Our morphology model is similar to that used by Goldsmith in his unsupervised morphological learning system [8], with two important differences. First, Goldsmith's model is recursive, i.e. a word stem can be further split into a smaller stem plus suffix. Second, Goldsmith's model assumes that all occurrences of each word type have the same analysis, whereas our model allows different tokens of the same type to have different analyses.

### 4.2 Inference by Gibbs sampling

Our goal in defining this morphology model is to be able to automatically infer the morphological structure of a language. This can be done using Gibbs sampling, a standard Markov chain Monte Carlo (MCMC) method [15]. In MCMC, variables in the model are repeatedly sampled, with each sample conditioned on the current values of all other variables in the model. This process defines a Markov chain whose stationary distribution is the posterior distribution over model variables given the input data.

Rather than sampling all the variables in our two-stage model simultaneously, our Gibbs sampler alternates between sampling the variables in the generator and those in the adaptor. Fixing the assignment of words to tables, we sample $c_k$, $t_k$, and $f_k$ for each table from

$$
\begin{aligned}
P(c_k = c, &t_k = t, f_k = f \mid \mathbf{c}_{-k}, \mathbf{t}_{-k}, \mathbf{f}_{-k}, \boldsymbol{\ell}) \\
\propto \quad &I(\ell_k = t_k.f_k) \quad P(c_k = c \mid \mathbf{c}_{-k}) \quad P(t_k = t \mid \mathbf{t}_{-k}, \mathbf{c}) \quad P(f_k = f \mid \mathbf{f}_{-k}, \mathbf{c}) \\
= \quad &I(\ell_k = t_k.f_k) \cdot \frac{n_c + \kappa}{K(\mathbf{z}) - 1 + \kappa C} \cdot \frac{n_{c,t} + \tau}{n_c + \tau T} \cdot \frac{n_{c,f} + \phi}{n_c + \phi F} \quad (8)
\end{aligned}
$$

where $n_c$ is the number of other labels assigned to class $c$, $n_{c,t}$ and $n_{c,f}$ are the number of other labels in class $c$ with stem $t$ and suffix $f$, respectively, and $C$, $T$, and $F$, are the total number of possible classes, stems, and suffixes, which are fixed. We use the notation $\mathbf{c}_{-k}$ here to indicate all members of $\mathbf{c}$ except for $c_k$. Equation 8 is obtained by integrating over the multinomial distributions specified in Equation 7, exploiting the conjugacy between multinomial and Dirichlet distributions.

Fixing the morphological analysis $(\mathbf{c}, \mathbf{t}, \mathbf{f})$, we sample the table $z_i$ for each word token from

$$
P(z_i = k \mid \mathbf{z}_{-i}, \mathbf{w}, \mathbf{c}, \mathbf{t}, \mathbf{f}) \propto
\begin{cases}
I(\ell_k = w_i)(n_k^{(\mathbf{z}_{-i})} - a) & n_k^{(\mathbf{z}_{-i})} > 0 \\
P(\ell_k = w_i)(K(\mathbf{z}_{-i})a + b) & n_k^{(\mathbf{z}_{-i})} = 0
\end{cases} \quad (9)
$$

where $P(\ell_k = w_i)$ is found using Equation 7, with $P(c)$, $P(t)$, and $P(f)$ replaced with the corresponding conditional distributions from Equation 8.

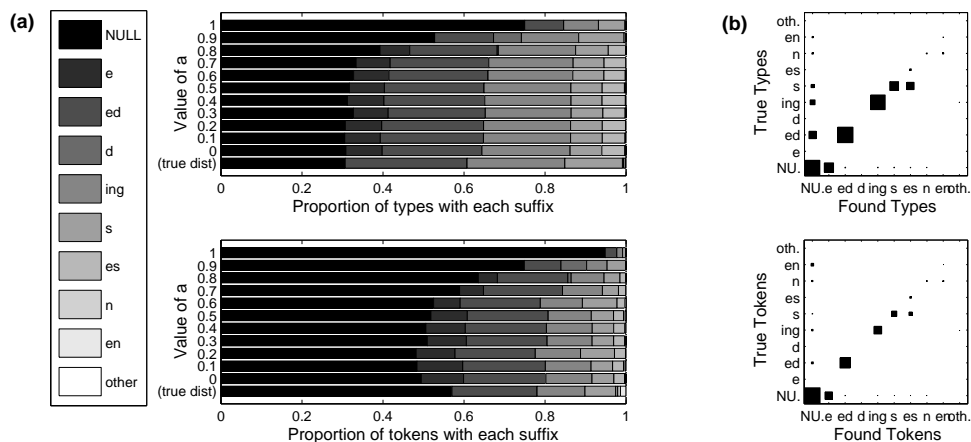

Figure 2: (a) Results for the morphology model, varying $a$. (b) Confusion matrices for the morphology model with $a = 0$. The area of a square at location $(i, j)$ is proportional to the number of word types (top) or tokens (bottom) with true suffix $i$ and found suffix $j$.

## 4.3 Experiments

We applied our model to a data set consisting of all the verbs in the training section of the Penn Wall Street Journal treebank (137,997 tokens belonging to 7,761 types). This simple test case using only a single part of speech makes our results easy to analyze. We determined the true suffix of each word using simple heuristics based on the part-of-speech tag and spelling of the word.[3] We then ran a Gibbs sampler using 6 classes, and compared the results of our learning algorithm to the true suffixes found in the corpus.

As noted above, the Gibbs sampler does not converge to a single analysis of the data, but rather to a distribution over analyses. For evaluation, we used a single sample taken after 1000 iterations. Figure 2 (a) shows the distribution of suffixes found by the model for various values of $a$, as well as the true distribution. We analyzed the results in two ways: by counting each suffix once for each word type it was associated with, and by counting once for each word token (thus giving more weight to the results for frequent words).

The most salient aspect of our results is that, regardless of whether we evaluate on types or tokens, it is clear that low values of $a$ are far more effective for learning morphology than higher values. With higher values of $a$, the system has too strong a preference for empty suffixes. This observation seems to support the linguists' view of type-based generalization.

It is also worth explaining why our morphological learner finds so many *e* and *es* suffixes. This problem is common to other morphological learning systems with similar models (e.g. [8]) and is due to the spelling rule in English that deletes stem-final *e* before certain suffixes. Since the system has no knowledge of spelling rules, it tends to hypothesize analyses such as {*stat.e, stat.ing, stat.ed, stat.es*}, where the *e* and *es* suffixes take the place of *NULL* and *s*. This effect can be seen clearly in the confusion matrices shown in Figure 2 (b). The remaining errors seen in the confusion matrices are those where the system hypothesized an empty suffix when in fact a non-empty suffix was present. Analysis of our results showed that these cases were mostly words where no other form with the same stem was present in

the corpus. There was therefore no reason for the system to prefer a non-empty suffix.

## 5  Conclusion

We have shown that statistical language models that exhibit one of the most striking properties of natural languages – power-law distributions – can be defined by breaking the process of generating words into two stages, with a generator producing a set of words, and an adaptor determining their frequencies. Our morphology model and the Pitman-Yor process are particular choices for a generator and an adaptor. These choices produce empirical and theoretical results that justify the role of word types in formal analyses of natural language. However, the greatest strength of this framework lies in its generality: we anticipate that other choices of generators and adaptors will yield similarly interesting results.

## Footnotes

[1] Under the interpretation of this model as a Pitman-Yor process mixture model, this is analogous to estimating the base measure $G_0$ in a Dirichlet process mixture model (e.g. [13]).

[2] Despite the fact that $P(\mathbf{w}\,|\,\theta)$ approaches 0 in this limit, $a^{K(\mathbf{z})-1}$ will be constant across all choices of $\theta$. Consequently, estimation schemes that depend only on the non-constant terms in $P(\mathbf{w}\,|\,\theta)$, such as maximum-likelihood or Bayesian inference, will remain well defined.

[3]The part-of-speech tags distinguish between past tense, past participle, progressive, 3rd person present singular, and infinitive/unmarked verbs, and therefore roughly correlate with actual suffixes.

## References

[1] G. Zipf. *Selective Studies and the Principle of Relative Frequency in Language.* Harvard University Press, Cambridge, MA, 1932.

[2] M. Mitzenmacher. A brief history of generative models for power law and lognormal distributions. *Internet Mathematics*, 1(2):226–251, 2003.

[3] H.A. Simon. On a class of skew distribution functions. *Biometrika*, 42(3/4):425–440, 1955.

[4] J. Pitman. Exchangeable and partially exchangeable random partitions. *Probability Theory and Related Fields*, 102:145–158, 1995.

[5] J. Pitman and M. Yor. The two-parameter Poisson-Dirichlet distribution derived from a stable subordinator. *Annals of Probability*, 25:855–900, 1997.

[6] H. Ishwaran and L. F. James. Generalized weighted Chinese restaurant processes for species sampling mixture models. *Statistica Sinica*, 13:1211–1235, 2003.

[7] J. B. Pierrehumbert. Probabilistic phonology: discrimination and robustness. In R. Bod, J. Hay, and S. Jannedy, editors, *Probabilistic linguistics*. MIT Press, Cambridge, MA, 2003.

[8] J. Goldsmith. Unsupervised learning of the morphology of a natural language. *Computational Linguistics*, 27:153–198, 2001.

[9] H. Ney, U. Essen, and R. Kneser. On structuring probabilistic dependences in stochastic language modeling. *Computer, Speech, and Language*, 8:1–38, 1994.

[10] R. Kneser and H. Ney. Improved backing-off for $n$-gram language modeling. In *Proceedings of the IEEE International Conference on Acoustics, Speech and Signal Processing*, 1995.

[11] S. F. Chen and J. Goodman. An empirical study of smoothing techniques for language modeling. Technical Report TR-10-98, Center for Research in Computing Technology, Harvard University, 1998.

[12] D. Aldous. Exchangeability and related topics. In *École d'été de probabilités de Saint-Flour, XIII—1983*, pages 1–198. Springer, Berlin, 1985.

[13] R. M. Neal. Markov chain sampling methods for Dirichlet process mixture models. *Journal of Computational and Graphical Statistics*, 9:249–265, 2000.

[14] L. Larkey, L. Ballesteros, and M. Connell. Improving stemming for arabic information retrieval: Light stemming and co-occurrence analysis. In *Proceedings of the 25th International Conference on Research and Development in Information Retrieval (SIGIR)*, 2002.

[15] W.R. Gilks, S. Richardson, and D. J. Spiegelhalter, editors. *Markov Chain Monte Carlo in Practice*. Chapman and Hall, Suffolk, 1996.
